# GENESIS: A SYSTEM FOR SIMULATING NEURAL NETWORKS

Matthew A. Wilson, Upinder S. Bhalla, John D. Uhley, James M. Bower.
Division of Biology
California Institute of Technology
Pasadena, CA 91125

## ABSTRACT

We have developed a graphically oriented, general purpose simulation system to facilitate the modeling of neural networks. The simulator is implemented under UNIX and X-windows and is designed to support simulations at many levels of detail. Specifically, it is intended for use in both applied network modeling and in the simulation of detailed, realistic, biologically-based models. Examples of current models developed under this system include mammalian olfactory bulb and cortex, invertebrate central pattern generators, as well as more abstract connectionist simulations.

## INTRODUCTION

Recently, there has been a dramatic increase in interest in exploring the computational properties of networks of parallel distributed processing elements (Rumelhart and McClelland, 1986) often referred to as "neural networks" (Anderson, 1988). Much of the current research involves numerical simulations of these types of networks (Anderson, 1988; Touretzky, 1989). Over the last several years, there has also been a significant increase in interest in using similar computer simulation techniques to study the structure and function of biological neural networks. This effort can be seen as an attempt to reverse-engineer the brain with the objective of understanding the functional organization of its very complicated networks (Bower, 1989). Simulations of these systems range from detailed reconstructions of single neurons, or even components of single neurons, to simulations of large networks of complex neurons (Koch and Segev, 1989). Modelers associated with each area of research are likely to benefit from exposure to a large range of neural network simulations. A simulation package capable of implementing these varied types of network models would facilitate this interaction.

## DESIGN FEATURES OF THE SIMULATOR

We have built GENESIS (GEneral NEtwork SImulation System) and its graphical interface XODUS (X-based Output and Display Utility for Simulators) to provide a standardized and flexible means of constructing neural network simulations while making minimal assumptions about the actual structure of the neural components. The system is capable of growing according to the needs of users by incorporating user-defined code. We will now describe the specific features of this system.

### Device independence.

The entire system has been designed to run under UNIX and X-windows (version 11) for maximum portability. The code was developed on Sun workstations and has been ported to Sun3's, Sun4's, Sun 386i's, and Masscomp computers. It should be portable to all installations supporting UNIX and X-11. In addition, we will be developing a parallel implementation of the simulation system (Nelson et al., 1989).

### Modular design.

The design of the simulator and interface is based on a "building-block" approach. Simulations are constructed of modules which receive inputs, perform calculations on them, and generate outputs (figs. 2,3). This approach is central to the generality and flexibility of the system as it allows the user to easily add new features without modification to the base code.

### Interactive specification and control.

Network specification and control is done at a high level using graphical tools and a network specification language (fig. 1). The graphics interface provides the highest and most user friendly level of interaction. It consists of a number of tools which the user can configure to suit a particular simulation. Through the graphical interface the user can display, control and adjust the parameters of simulations. The network specification language we have developed for network modeling represents a more basic level of interaction. This language consists of a set of simulator and interface functions that can be executed interactively from the keyboard or from text files storing command sequences (scripts). The language also provides for arithmetic operations and program control functions such as looping, conditional statements, and subprograms or macros. Figures 3 and 4 demonstrate how some of these script functions are used.

### Simulator and interface toolkits.

Extendable toolkits which consist of module libraries, graphical tools and the simulator base code itself (fig. 2) provide the routines and modules used to construct specific simulations. The base code provides the common control and support routines for the entire system.

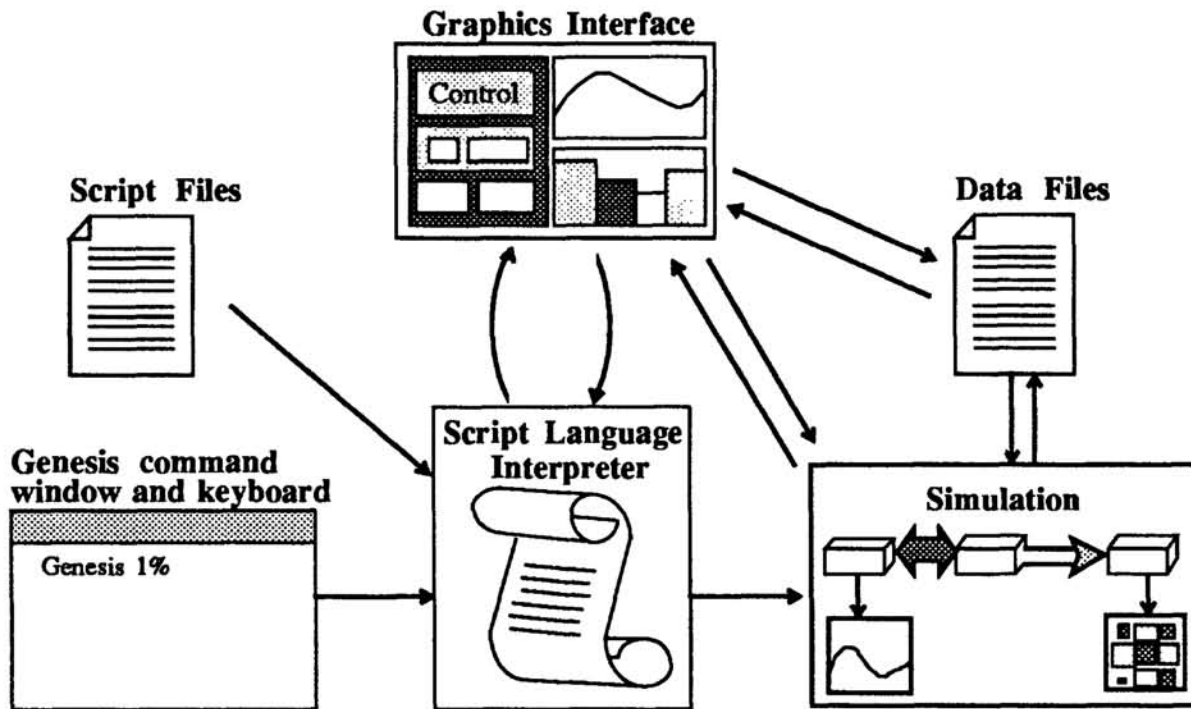

**Figure 1.** Levels Of Interaction With The Simulator

# CONSTRUCTING SIMULATIONS

The first step in using GENESIS involves selecting and linking together those modules from the toolkits that will be necessary for a particular simulation (fig. 2,3). Additional commands in the scripting language establish the network and the graphical interface (fig. 4).

**Module Classes.**

Modules in GENESIS are divided into computational modules, communications modules and graphical modules. All instances of computational modules are called *elements*. These are the central components of simulations, performing all of the numerical calculations. Elements can communicate in two ways: via *links* and via *connections*. Links allow the passing of data between two elements with no time delay and with no computation being performed on the data. Thus, links serve to unify a large number of elements into a single computational unit (e.g. they are used to link elements together to form the neuron in fig. 3C). Connections, on the other hand, interconnect computational units via simulated communication channels which can incorporate time delays and perform transformations on data being transmitted (e.g. axons in fig. 3C). Graphical modules called *widgets* are used to construct the interface. These modules can issue script commands as well as respond to them, thus allowing interactive access to simulator structures and functions.

**Hierarchical organization.**

In order to keep track of the structure of a simulation, elements are organized into a tree hierarchy similar to the directory structure in UNIX (fig. 3B). The tree structure does not explicitly represent the pattern of links and connections between elements, it is simply a tool for organizing complex groups of elements in the simulation.

**Simulation example.**

As an example of the types of modules available and the process of structuring them into a network simulation and graphical interface, we will describe the construction of a simple biological neural simulation (fig. 3). The model consists of two neurons. Each neuron contains a passive dendritic compartment, an active cell body, an axonal output, and a synaptic input onto the dendrite. The axon of one neuron connects to a synaptic input of the other. Figure 3 shows the basic structure of the model as implemented under GENESIS. In the model, the synapse, channels,

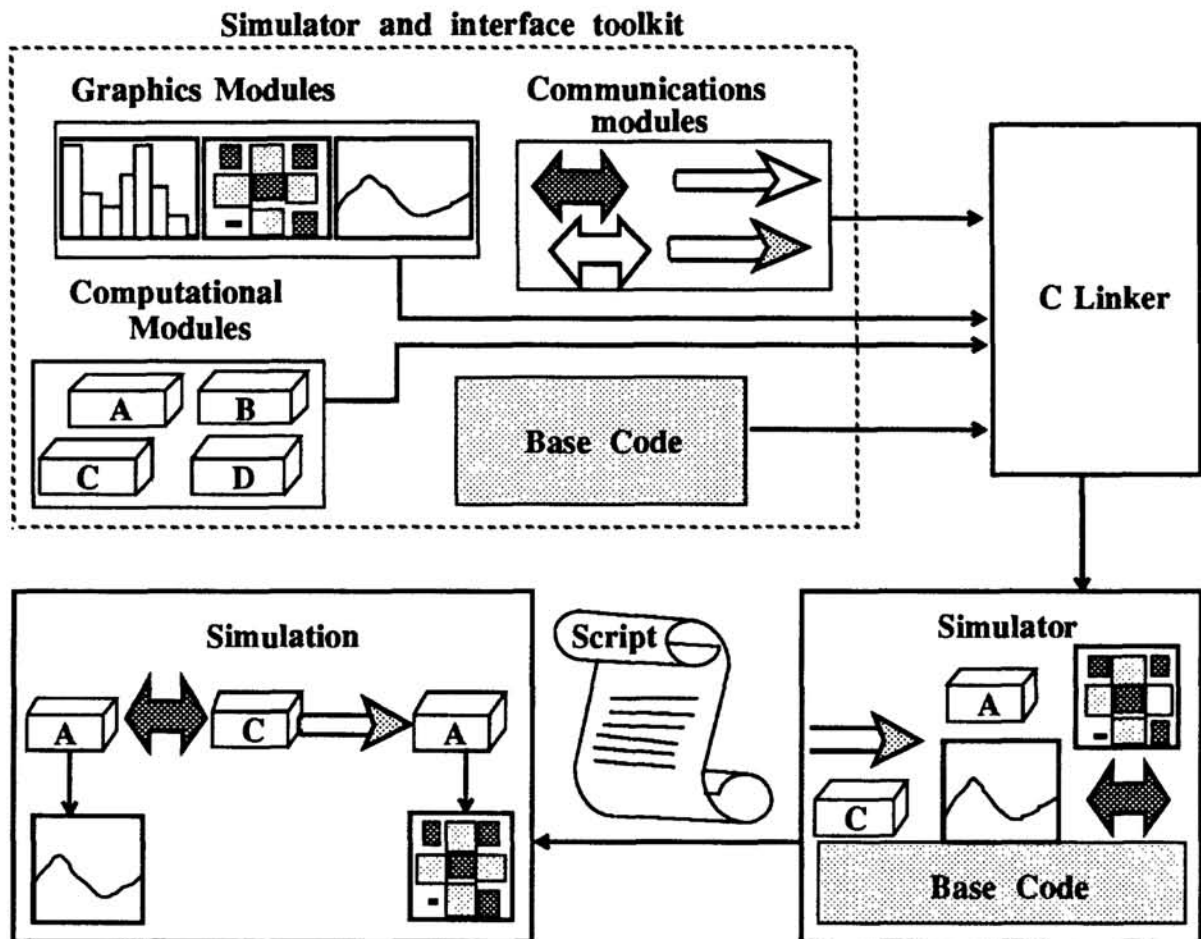

**Figure 2.** Stages In Constructing A Simulation.

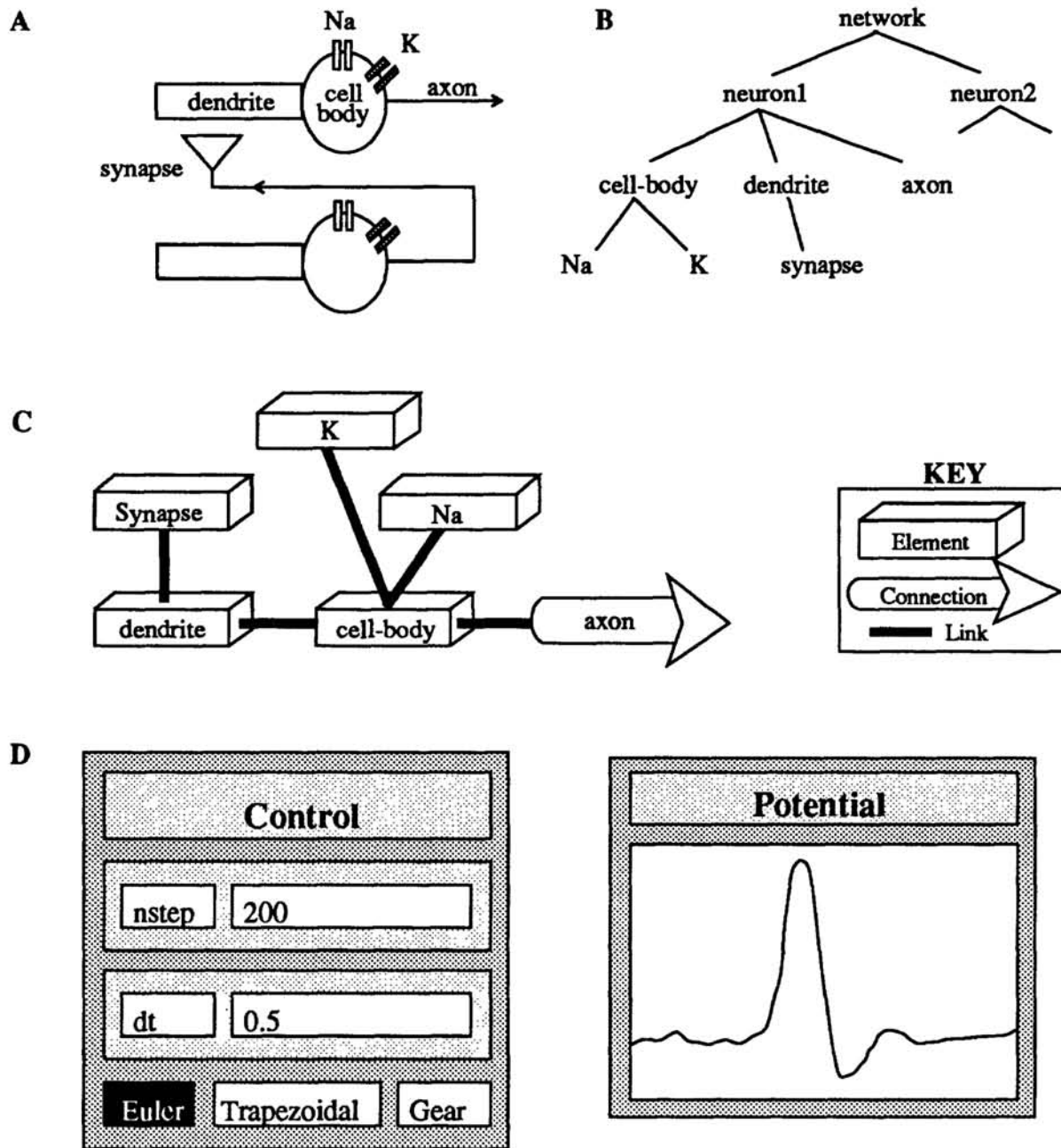

**Figure 3.** Implementation of a two neuron model in GENESIS. **(A)** Schematic diagram of compartmentally modeled neurons. Each cell in this simple model has a passive dendritic compartment, an active cell-body, and an output axon. There is a synaptic input to the dendrite of one cell and two ionic channels on the cell body. **(B)** Hierarchical representation of the components of the simulation as maintained in GENESIS. The cell-body  of neuron 1 is referred to as /network/neuron1/cell-body. **(C)** A representation of the functional links between the basic components of one neuron. **(D)** Sample interface control and display widgets created using the XODUS toolkit.

dendritic compartments, cell body and axon are each treated as separate computational elements (fig. 3C). Links allow elements to share information (e.g. the Na channel needs to have access to the cell-body membrane voltage). Figure 4 shows a portion of the script used to construct this simulation.

---

**Create different types of elements and assign them names.**

```
create                            neuron1
create    active_compartment      cell-body
create    passive_compartment     dendrite
create    synapse                 dendrite/synapse
```

**Establish functional "links" between the elements.**

```
link     dendrite           to     cell-body
link     dendrite/synapse   to     dendrite
```

**Set parameters associated with the elements.**

```
set      dendrite    capacitance    1.0e-6
```

**Make copies of entire element subtrees.**

```
copy     neuron1            to     neuron2
```

**Establish "connections" between two elements.**

```
connect  neuron1/axon       to     neuron2/dendrite/synapse
```

**Set up a graph to monitor an element variable**

```
graph    neuron1/cell-body          potential
```

**Make a control panel with several control "widgets".**

```
xform    control
xdialog  nstep    set-nstep   -default 200
xdialog  dt       set-dt      -default 0.5
xtoggle  Euler    set-euler
```

**Figure 4.** Sample script commands for constructing a simulation (see fig. 3)

---

## SIMULATOR SPECIFICATIONS

**Memory requirements of GENESIS.**

Currently, GENESIS consists of about 20,000 lines of simulator code and a similar amount of graphics code, all written in C. The executable binaries take up about 1.5 Megabytes. A rough estimate of the amount of additional memory necessary for a particular simulation can be calculated from the sizes and number of modules used in a simulation. Typically, elements use around 100 bytes, connections 16 and messages 20. Widgets use 5-20 Kbytes each.

## Performance

The overall efficiency of the GENESIS system is highly simulation specific. To consider briefly a specific case, the most sophisticated biologically based simulation currently implemented under GENESIS, is a model of piriform (olfactory) cortex (Wilson et al., 1986; Wilson and Bower, 1988; Wilson and Bower, 1989). This simulation consists of neurons of four different types. Each neuron contains from one to five compartments. Each compartment can contain several channels. On a SUN 386i with 8 Mbytes of RAM, this simulation with 500 cells runs at 1 second per time step.

## Other models that have been implemented under GENESIS

The list of projects currently completed under GENESIS includes approximately ten different simulations. These include models of the olfactory bulb (Bhalla et al., 1988), the inferior olive (Lee and Bower, 1988), and a motor circuit in the invertebrate sea slug Tritonia (Ryckebusch et al., 1989). We have also built several tutorials to allow students to explore compartmental biological models (Hodgkin and Huxley, 1952), and Hopfield networks (Hopfield, 1982).

## Access/use of GENESIS

GENESIS and XODUS will be made available at the cost of distribution to all interested users. As described above, new user-defined modules can be linked into the simulator to extend the system. Users are encouraged to support the continuing development of this system by sending modules they develop to Caltech. These will be reviewed and compiled into the overall system by GENESIS support staff. We would also hope that users would send completed published simulations to the GENESIS data base. This will provide others with an opportunity to observe the behavior of a simulation first hand. A current listing of modules and full simulations will be maintained and available through an electronic mail newsgroup, Babel. Enquiries about the system should be sent to GENESIS@caltech.edu or GENESIS@caltech.bitnet.

## Acknowledgments

We would like to thank Mark Nelson for his invaluable assistance in the development of this system and specifically for his suggestions on the content of this manuscript. We would also like to recognize Dave Bilitch, Wojtek Furmanski, Christof Koch, innumerable Caltech students and the students of the 1988 MBL summer course on Methods in Computational Neuroscience for their contributions to the creation and evolution of GENESIS (not mutually exclusive). This research was also supported by the NSF (EET-8700064), the NIH (BNS 22205), the ONR (Contract N00014-88-K-0513), the Lockheed Corporation, the Caltech Presidents Fund, the JPL Directors Development Fund, and the Joseph Drown Foundation.

## References

D. Anderson. (ed.) Neural information processing systems. American Institute of Physics, New York (1988).

U.S. Bhalla, M.A. Wilson, & J.M. Bower. Integration of computer simulations and multi-unit recording in the rat olfactory system. Soc. Neurosci. Abstr. 14: 1188 (1988).

J.M. Bower. Reverse engineering the nervous system: An anatomical, physiological, and computer based approach. In: An Introduction to Neural and Electronic Networks. Zornetzer, Davis, and Lau, editors. Academic Press (1989)(in press).

A.L. Hodgkin and A.F. Huxley. A quantitative description of membrane current and its application to conduction and excitation in nerve. J.Physiol, (Lond.) 117, 500-544 (1952).

J.J. Hopfield. Neural networks and physical systems with emergent collective computational abilities. Proc. Natl. Acad. Sci. USA. 79, 2554-2558 (1982).

C. Koch and I. Segev. (eds.) Methods in Neuronal Modeling: From Synapses to Networks. MIT Press, Cambridge, MA (in press).

M. Lee and J.M. Bower. A structural simulation of the inferior olivary nucleus. Soc. Neurosci. Abstr. 14: 184 (1988).

M. Nelson, W. Furmanski and J.M. Bower. Simulating neurons and neuronal networks on parallel computers. In: Methods in Neuronal Modeling: From Synapses to Networks. C. Koch and I. Segev, editors. MIT Press, Cambridge, MA (1989)(in press).

S. Ryckebusch, C. Mead and J.M. Bower. Modeling a central pattern generator in software and hardware: Tritonia in sea moss (CMOS). (1989)(this volume).

D.E. Rumelhart, J.L. McClelland and the PDP Research Group. Parallel Distributed Processing. MIT Press, Cambridge, MA (1986).

D. Touretzky. (ed.) Advances in Neural Network Information Processing Systems. Morgan Kaufmann Publishers, San Mateo, California (1989).

M.A. Wilson and J.M. Bower. The simulation of large-scale neuronal networks. In: Methods in Neuronal Modeling: From Synapses to Networks. C. Koch and I. Segev, editors. MIT Press, Cambridge, MA (1989)(in press).

M.A. Wilson and J.M. Bower. A computer simulation of olfactory cortex with functional implications for storage and retrieval of olfactory information. In: Neural information processing systems. pp. 114-126 D. Anderson, editor. Published by AIP Press, New York, N.Y (1988).

M.A. Wilson, J.M. Bower and L.B. Haberly. A computer simulation of piriform cortex. Soc. Neurosci. Abstr. 12,1358 (1986).


.

.